# Semidefinite relaxations for approximate inference on graphs with cycles

**Martin J. Wainwright**
Electrical Engineering and Computer Science
UC Berkeley, Berkeley, CA 94720
`wainwrig@eecs.berkeley.edu`

**Michael I. Jordan**
Computer Science and Statistics
UC Berkeley, Berkeley, CA 94720
`jordan@cs.berkeley.edu`

## Abstract

We present a new method for calculating approximate marginals for probability distributions defined by graphs with cycles, based on a Gaussian entropy bound combined with a semidefinite outer bound on the marginal polytope. This combination leads to a log-determinant maximization problem that can be solved by efficient interior point methods [8]. As with the Bethe approximation and its generalizations [12], the optimizing arguments of this problem can be taken as approximations to the exact marginals. In contrast to Bethe/Kikuchi approaches, our variational problem is strictly convex and so has a unique global optimum. An additional desirable feature is that the value of the optimal solution is guaranteed to provide an upper bound on the log partition function. In experimental trials, the performance of the log-determinant relaxation is comparable to or better than the sum-product algorithm, and by a substantial margin for certain problem classes. Finally, the zero-temperature limit of our log-determinant relaxation recovers a class of well-known semidefinite relaxations for integer programming [e.g., 3].

## 1 Introduction

Given a probability distribution defined by a graphical model (e.g., Markov random field, factor graph), a key problem is the computation of marginal distributions. Although highly efficient algorithms exist for trees, exact solutions are prohibitively complex for more general graphs of any substantial size. This difficulty motivates the use of algorithms for computing approximations to marginal distributions, a problem to which we refer as *approximate inference*. One widely-used algorithm is the belief propagation or sum-product algorithm. As shown by Yedidia et al. [12], it can be interpreted as a method for attempting to solve a variational problem wherein the exact entropy is replaced by the Bethe approximation. Moreover, Yedidia et al. proposed extensions to the Bethe approximation based on clustering operations.

An unattractive feature of the Bethe approach and its extensions is that with certain exceptions [e.g., 6], the associated variational problems are typically not convex, thus leading to algorithmic complications, and also raising the possibility of multiple local optima. Secondly, in contrast to other variational methods (e.g., mean field [4]), the optimal values of Bethe-type variational problems fail to provide bounds on the log partition function. This

function arises in various contexts, including approximate parameter estimation and large deviations exponents, so that such bounds are of interest in their own right.

This paper introduces a new class of variational problems that are both convex and provide upper bounds. Our derivation relies on a Gaussian upper bound on the discrete entropy of a suitably regularized random vector, and a semidefinite outer bound on the set of valid marginal distributions. The combination leads to a log-determinant maximization problem with a unique optimum that can be found by efficient interior point methods [8]. As with the Bethe/Kikuchi approximations and sum-product algorithms, the optimizing arguments of this problem can be taken as approximations to the marginal distributions of the underlying graphical model. Moreover, taking the "zero-temperature" limit recovers a class of well-known semidefinite programming relaxations for integer programming problems [e.g., 3].

## 2   Problem set-up

We consider an undirected graph $G = (V, E)$ with $n = |V|$ nodes. Associated with each vertex $s \in V$ is a random variable $x_s$ taking values in the discrete space $\mathcal{X} = \{0, 1, \ldots, m - 1\}$. We let $\mathbf{x} = \{x_s \mid s \in V\}$ denote a random vector taking values in the Cartesian product space $\mathcal{X}^n$. Our analysis makes use of the following exponential representation of a graph-structured distribution $p(\mathbf{x})$. For some index set $\mathcal{I}$, we let $\phi = \{\phi_\alpha \mid \alpha \in \mathcal{I}\}$ denote a collection of potential functions associated with the cliques of $G$, and let $\theta = \{\theta_\alpha \mid \alpha \in \mathcal{I}\}$ be a vector of parameters associated with these potential functions. The exponential family determined by $\phi$ is the following collection:

$$p(\mathbf{x}; \theta) = \exp\Big\{\sum_\alpha \theta_\alpha \phi_\alpha(\mathbf{x}) - \Phi(\theta)\Big\} \tag{1a}$$

$$\Phi(\theta) = \log \sum_{\mathbf{x} \in \mathcal{X}^n} \exp\Big\{\sum_\alpha \theta_\alpha \phi_\alpha(\mathbf{x})\Big\}. \tag{1b}$$

Here $\Phi(\theta)$ is the *log partition function* that serves to normalize the distribution. In a *minimal* representation, the functions $\{\phi_\alpha\}$ are affinely independent, and $d = |\mathcal{I}|$ corresponds to the dimension of the family. For example, one minimal representation of a binary-valued random vector on a graph with pairwise cliques is the standard Ising model, in which $\phi = \{x_s \mid s \in V\} \cup \{x_s x_t \mid (s, t) \in E\}$. Here the index set $\mathcal{I} = V \cup E$, and $d = n + |E|$. In order to incorporate higher order interactions, we simply add higher degree monomials (e.g., $x_s x_t x_u$ for a third order interaction) to the collection of potential functions. Similar representations exist for discrete processes on alphabets with $m > 2$ elements.

### 2.1   Duality and marginal polytopes

It is well known that $\Phi$ is convex in terms of $\theta$, and strictly so for a minimal representation. Accordingly, it is natural to consider its conjugate dual function, which is defined by the relation:

$$\Phi^*(\mu) = \sup_{\theta \in \mathbb{R}^d} \{\langle \mu, \theta \rangle - \Phi(\theta)\}. \tag{2}$$

Here the vector of *dual variables* $\mu$ is the same dimension as exponential parameter $\theta$ (i.e., $\mu \in \mathbb{R}^d$). It is straightforward to show that the partial derivatives of $\Phi$ with respect to $\theta$ correspond to cumulants of $\phi(\mathbf{x})$; in particular, the first order derivatives define mean parameters:

$$\frac{\partial \Phi}{\partial \theta_\alpha}(\theta) = \sum_{\mathbf{x} \in \mathcal{X}^n} p(\mathbf{x}; \theta)\phi_\alpha(\mathbf{x}) = \mathbb{E}_\theta[\phi_\alpha(\mathbf{x})]. \tag{3}$$

In order to compute $\Phi^*(\mu)$ for a given $\mu$, we take the derivative with respect to $\theta$ of the quantity within curly braces in Eqn. (2). Setting this derivative to zero and making use of Eqn. (3) yields defining conditions for a vector $\theta(\mu)$ attaining the optimum in Eqn. (2):

$$\mu_\alpha \;=\; \mathbb{E}_{\theta(\mu)}[\phi_\alpha(\mathbf{x})] \qquad \forall\, \alpha \in \mathcal{I} \tag{4}$$

It can be shown [10] that Eqn. (4) has a solution if and only if $\mu$ belongs to the relative interior of the set:

$$\mathrm{MARG}(G; \phi) \;=\; \{\, \mu \in \mathbb{R}^d \;\mid\; \sum_{\mathbf{x} \in \mathcal{X}^n} p(\mathbf{x})\, \phi(\mathbf{x}) = \mu \;\; \text{for} \;\; \text{some} \;\; p(\cdot)\} \tag{5}$$

Note that this set is equivalent to the convex hull of the finite collection of vectors $\{\phi(\mathbf{x}) \mid \mathbf{x} \in \mathcal{X}^n\}$; consequently, the Minkowski-Weyl theorem [7] guarantees that it can be characterized by a finite number of linear inequality constraints. We refer to this set as the *marginal polytope*[1] associated with the graph $G$ and the potentials $\phi$.

In order to calculate an explicit form for $\Phi^*(\mu)$ for any $\mu \in \mathrm{MARG}(G; \phi)$, we substitute the relation in Eqn. (4) into the definition of $\Phi^*$, thereby obtaining:

$$\Phi^*(\mu) \;=\; \langle \mu,\, \theta(\mu) \rangle - \Phi(\theta(\mu)) \;=\; \sum_{\mathbf{x} \in \mathcal{X}^n} p(\mathbf{x}; \theta(\mu)) \log p(\mathbf{x}; \theta(\mu)). \tag{6}$$

This relation establishes that for $\mu$ in the relative interior of $\mathrm{MARG}(G; \phi)$, the value of the conjugate dual $\Phi^*(\mu)$ is given by the negative entropy of the distribution $p(\mathbf{x}; \theta(\mu))$, where the pair $\theta(\mu)$ and $\mu$ are dually coupled via Eqn. (4). For $\mu \notin \mathrm{cl}\,\mathrm{MARG}(G; \phi)$, it can be shown [10] that the value of the dual is $+\infty$.

Since $\Phi$ is lower semi-continuous, taking the conjugate twice recovers the original function [7]; applying this fact to $\Phi^*$ and $\Phi$, we obtain the following relation:

$$\Phi(\theta) \;=\; \max_{\mu \in \mathrm{MARG}(G; \phi)} \{\langle \theta,\, \mu \rangle - \Phi^*(\mu)\}. \tag{7}$$

Moreover, we are guaranteed that the optimum is attained uniquely at the exact marginals $\mu = \{\mu_\alpha\}$ of $p(\mathbf{x}; \theta)$. This variational formulation plays a central role in our development in the sequel.

## 2.2 Challenges with the variational formulation

There are two difficulties associated with the variational formulation (7). First of all, observe that the (negative) entropy $\Phi^*$, as a function of *only* the mean parameters $\mu$, is implicitly defined; indeed, it is typically impossible to specify an explicit form for $\Phi^*$. Key exceptions are trees and hypertrees, for which the entropy is well-known to decompose into a sum of local entropies defined by local marginals on the (hyper)edges [1]. Secondly, for a general graph with cycles, the marginal polytope $\mathrm{MARG}(G; \phi)$ is defined by a number of inequalities that grows rapidly in graph size [e.g., 2]. Trees and hypertrees again are important exceptions: in this case, the junction tree theorem [e.g., 1] provides a compact representation of the associated marginal polytopes.

The Bethe approach (and its generalizations) can be understood as consisting of two steps: (a) replacing the exact entropy $-\Phi^*$ with a tree (or hypertree) approximation; and (b) replacing the marginal polytope $\mathrm{MARG}(G; \phi)$ with constraint sets defined by tree (or hypertree) consistency conditions. However, since the (hyper)tree approximations used do not bound the exact entropy, the optimal values of Bethe-type variational problems do not provide a bound on the value of the log partition function $\Phi(\theta)$. Requirements for bounding $\Phi$ are both an *outer bound* on the marginal polytope, as well as an *upper bound* on the entropy $-\Phi^*$.

# 3 Log-determinant relaxation

In this section, we state and prove a set of upper bounds based on the solution of a variational problem involving determinant maximization and semidefinite constraints. Although the ideas and methods described here are more generally applicable, for the sake of clarity in exposition we focus here on the case of a binary vector $\mathbf{x} \in \{-1, +1\}^n$ of "spins". It is also convenient to define all problems with respect to the complete graph $K_n$ (i.e., fully connected). We use the standard (minimal) Ising representation for a binary problem, in terms of the potential functions $\phi = \{x_s \mid s \in V\} \cup \{x_s x_t \mid (s,t)\}$. On the complete graph, there are $d = n + \binom{n}{2}$ such potential functions in total. Of course, any problem can be embedded into the complete graph by setting to zero a subset of the $\{\theta_{st}\}$ parameters. (In particular, for a graph $G = (V, E)$, we simply set $\theta_{st} = 0$ for all pairs $(s,t) \notin E$.)

## 3.1 Outer bounds on the marginal polytope

We first focus on the marginal polytope $\mathrm{MARG}(K_n) \equiv \mathrm{MARG}(K_n; \phi)$ of valid dual variables $\{\mu_s, \mu_{st}\}$, as defined in Eqn. (5). In this section, we describe a set of semidefinite and linear constraints that any valid dual vector $\mu \in \mathrm{MARG}(K_n)$ must satisfy.

### 3.1.1 Semidefinite constraints

Given an arbitrary vector $\mu \in \mathbb{R}^d$, consider the following $(n+1) \times (n+1)$ matrix:

$$M_1[\mu] \quad := \quad \begin{bmatrix} 1 & \mu_1 & \mu_2 & \cdots & \mu_{n-1} & \mu_n \\ \mu_1 & 1 & \mu_{12} & \cdots & \cdots & \mu_{1n} \\ \mu_2 & \mu_{21} & 1 & \cdots & \cdots & \mu_{2n} \\ \vdots & \vdots & \vdots & \vdots & \vdots & \vdots \\ \mu_{n-1} & \vdots & \vdots & \vdots & \vdots & \mu_{n,(n-1)} \\ \mu_n & \mu_{n1} & \mu_{n2} & \cdots & \mu_{(n-1),n} & 1 \end{bmatrix} \tag{8}$$

The motivation underlying this definition is the following: suppose that the given dual vector $\mu$ actually belongs to $\mathrm{MARG}(K_n)$, in which case there exists some distribution $p(\mathbf{x}; \theta)$ such that $\mu_s = \sum_{\mathbf{x}} p(\mathbf{x}; \theta) \, x_s$ and $\mu_{st} = \sum_{\mathbf{x}} p(\mathbf{x}; \theta) \, x_s x_t$. Thus, if $\mu \in \mathrm{MARG}(K_n)$, the matrix $M_1[\mu]$ can be interpreted as the matrix of second order moments for the vector $(1, \mathbf{x})$, as computed under $p(\mathbf{x}; \theta)$. (Note in particular that the diagonal elements are all one, since $x_s^2 = 1$ when $x_s \in \{-1, +1\}$.) Since any such moment matrix must be positive semidefinite,[2] we have established the following:

**Lemma 1 (Semidefinite outer bound).** *The binary marginal polytope* $\mathrm{MARG}(K_n)$ *is contained within the semidefinite constraint set:*

$$\mathrm{SDEF}_1 \quad := \quad \{ \mu \in \mathbb{R}^d \mid M_1[\mu] \succeq 0 \} \tag{9}$$

This semidefinite relaxation can be further strengthened by including higher order terms in the moment matrices [5].

### 3.1.2 Additional linear constraints

It is straightforward to augment these semidefinite constraints with additional linear constraints. Here we focus in particular on two classes of constraints, referred to as rooted and unrooted triangle inequalities by Deza and Laurent [2], that are of especial relevance in the graphical model setting.

**Pairwise edge constraints:** It is natural to require that the subset of mean parameters associated with each pair of random variables $(x_s, x_t)$ — namely, $\mu_s$, $\mu_t$ and $\mu_{st}$ — specify a valid pairwise marginal distribution. Letting $\{a, b\}$ take values in $\{-1, +1\}^2$, consider the set of four linear constraints of the following form:

$$1 + a\,\mu_s + b\,\mu_t + ab\,\mu_{st} \quad \geq \quad 0. \tag{10}$$

It can be shown [11, 10] that these constraints are necessary and sufficient to guarantee the existence of a consistent pairwise marginal. By the junction tree theorem [1], this pairwise consistency guarantees that the constraints of Eqn. (10) provide a complete description of the binary marginal polytope for any tree-structured graph. Moreover, for a general graph with cycles, they are equivalent to the tree-consistent constraint set used in the Bethe approach [12] when applied to a binary vector $\mathbf{x} \in \{-1, +1\}^n$.

**Triplet constraints:** Local consistency can be extended to triplets $\{x_s, x_t, x_u\}$, and even more generally to higher order subsets. For the triplet case, consider the following set of constraints (and permutations thereof) among the pairwise mean parameters $\{\mu_{st}, \mu_{su}, \mu_{tu}\}$:

$$\mu_{st} + \mu_{su} + \mu_{tu} \geq -1, \qquad \mu_{st} - \mu_{su} - \mu_{tu} \geq -1. \tag{11}$$

It can be shown [11, 10] that these constraints, in conjunction with the pairwise constraints (10), are necessary and sufficient to ensure that the collection of mean parameters $\{\mu_s, \mu_t, \mu_u, \mu_{st}, \mu_{su}, \mu_{tu}\}$ uniquely determine a valid marginal over the triplet $(x_s, x_t, x_u)$. Once again, by applying the junction tree theorem [1], we conclude that the constraints (10) and (11) provide a complete characterization of the binary marginal polytope for hypertrees of width two. It is worthwhile observing that this set of constraints is equivalent to those that are implicitly enforced by any Kikuchi approximation [12] with clusters of size three (when applied to a binary problem).

### 3.2  Gaussian entropy bound

We now turn to the task of upper bounding the entropy. Our starting point is the familiar interpretation of the Gaussian as the maximum entropy distribution subject to covariance constraints:

**Lemma 2.** *The (differential) entropy $h(\widetilde{\mathbf{x}}) := -\int p(\widetilde{\mathbf{x}}) \log p(\widetilde{\mathbf{x}}) d\widetilde{\mathbf{x}}$ is upper bounded by the entropy $\frac{1}{2}\log \det \operatorname{cov}(\widetilde{\mathbf{x}}) + \frac{n}{2}\log(2\pi e)$ of a Gaussian with matched covariance.*

Of interest to us is the discrete entropy of a discrete-valued random vector $\mathbf{x} \in \{-1, +1\}^n$, whereas the Gaussian bound of Lemma 2 applies to the differential entropy of a continuous-valued random vector. Therefore, we need to convert our discrete vector to the continuous space. In order to do so, we define a new continuous random vector via $\widetilde{\mathbf{x}} = \frac{1}{2}\mathbf{x} + \mathbf{u}$, where $\mathbf{u}$ is a random vector independent of $\mathbf{x}$, with each element independently and identically distributed[3] as $u_s \sim \mathcal{U}[-\frac{1}{2}, \frac{1}{2}]$. The motivation for rescaling $\mathbf{x}$ by $\frac{1}{2}$ is to pack the boxes together as tightly as possible.

**Lemma 3.** *We have $h(\widetilde{\mathbf{x}}) = H(\mathbf{x})$, where $h$ and $H$ denote the differential and discrete entropies of $\widetilde{\mathbf{x}}$ and $\mathbf{x}$ respectively.*

*Proof.* By construction, the differential entropy can be decomposed as a sum of integrals over hyperboxes of unit volume, one for each configuration, over which the probability density of $\widetilde{\mathbf{x}}$ is constant. ∎

### 3.3  Log-determinant relaxation

Equipped with these building blocks, we are now ready to state and prove a log-determinant relaxation for the log partition function.

**Theorem 1.** *Let* $\mathbf{x} \in \{-1, +1\}^n$, *and let* $\mathrm{OUT}(K_n)$ *be any convex outer bound on* $\mathrm{MARG}(K_n)$ *that is contained within* $\mathrm{SDEF}_1$. *Then there holds*

$$\Phi(\theta) \; \leq \; \max_{\mu \in \mathrm{OUT}(K_n)} \left\{ \langle \theta, \, \mu \rangle + \frac{1}{2} \log \det \left[ M_1(\mu) + \frac{1}{3} \, \mathrm{blkdiag}[0, I_n] \right] \right\} + \frac{n}{2} \log(\frac{\pi e}{2}) \tag{12}$$

*where* $\mathrm{blkdiag}[0, I_n]$ *is an* $(n+1) \times (n+1)$ *block-diagonal matrix. Moreover, the optimum is attained at a unique* $\widehat{\mu} \in \mathrm{OUT}(K_n)$.

*Proof.* For any $\mu \in \mathrm{MARG}(K_n)$, let $\mathbf{x}$ be a random vector with these mean parameters. Consider the continuous-valued random vector $\widetilde{\mathbf{x}} = \frac{1}{2}\mathbf{x} + \mathbf{u}$. From Lemma 3, we have $H(\mathbf{x}) = h(\widetilde{\mathbf{x}})$; combining this equality with Lemma 2, we obtain the upper bound $H(\mathbf{x}) \leq \frac{1}{2} \log \det \mathrm{cov}(\widetilde{\mathbf{x}}) + \frac{n}{2} \log(2\pi e)$. Since $\mathbf{x}$ and $\mathbf{u}$ are independent and $\mathbf{u} \sim \mathcal{U}[-1/2, 1/2]$, we can write $\mathrm{cov}(\widetilde{\mathbf{x}}) = \frac{1}{4} \mathrm{cov}(\mathbf{x}) + \frac{1}{12} I_n$. Next we use the Schur complement formula [8] to express the log determinant as follows:

$$\log \det \mathrm{cov}(\widetilde{\mathbf{x}}) \;\; = \;\; \log \det \left\{ M_1[\mu] + \frac{1}{3} \, \mathrm{blkdiag}[0, I_n] \right\} + n \log \frac{1}{4}. \tag{13}$$

Combining Eqn. (13) with the Gaussian upper bound leads to the following expression:

$$H(\mathbf{x}) \;\; = \;\; -\Phi^*(\mu) \;\; \leq \;\; \frac{1}{2} \log \det \left( M_1[\mu] + \frac{1}{3} \, \mathrm{blkdiag}[0, I_n] \right) + \frac{n}{2} \log(\frac{\pi e}{2})$$

Substituting this upper bound into the variational representation of Eqn. (7) and using the fact that $\mathrm{OUT}(K_n)$ is an outer bound on $\mathrm{MARG}(G)$ yields Eqn. (12). By construction, the cost function is strictly convex so that the optimum is unique.  $\square$

The inclusion $\mathrm{OUT}(K_n) \subseteq \mathrm{SDEF}_1$ in the statement of Theorem 1 guarantees that the matrix $M_1(\mu)$ will always be positive semidefinite. Importantly, the optimization problem in Eqn. (12) is a determinant maximization problem, for which efficient interior point methods have been developed [e.g., 8].

## 4  Experimental results

The relevance of the log-determinant relaxation for applications is two-fold: it provides upper bounds on the log partition function, and the maximizing arguments $\widehat{\mu} \in \mathrm{OUT}(K_n)$ of Eqn. (12) can be taken as approximations to the exact marginals of the distribution $p(\mathbf{x}; \theta)$. So as to test its performance in computing approximate marginals, we performed extensive experiments on the complete graph (fully connected) and the 2-D nearest-neighbor lattice model. We treated relatively small problems with 16 nodes so as to enable comparison to the exact answer. For any given trial, we specified the distribution $p(\mathbf{x}; \theta)$ by randomly choosing $\theta$ as follows. The single node parameters were chosen as $\theta_s \sim \mathcal{U}[-0.25, 0.25]$ independently[4] for each node. For a given coupling strength $d_{\mathrm{coup}} > 0$, we investigated three possible types of coupling: (a) for *repulsive* interactions, $\theta_{st} \sim \mathcal{U}[-2d_{\mathrm{coup}}, 0]$; (b) for *mixed* interactions, $\theta_{st} \sim \mathcal{U}[-d_{\mathrm{coup}}, +d_{\mathrm{coup}}]$; (c) for *attractive* interactions, $\theta_{st} \sim \mathcal{U}[0, 2d_{\mathrm{coup}}]$.

For each distribution $p(\mathbf{x}; \theta)$, we performed the following computations: (a) the exact marginal probability $p(x_s = 1; \theta)$ at each node; and (b) approximate marginals computed

from the Bethe approximation with the sum-product algorithm, or (c) log-determinant approximate marginals from Theorem 1 using the outer bound $\text{OUT}(K_n)$ given by the first semidefinite relaxation $\text{SDEF}_1$ in conjunction with the pairwise linear constraints in Eqn. (10). We computed the exact marginal values either by exhaustive summation (complete graph), or by the junction tree algorithm (lattices). We used the standard parallel message-passing form of the sum-product algorithm with a damping factor[5] $\gamma = 0.05$. The log-determinant problem of Theorem 1 was solved using interior point methods [8]. For each graph (fully connected or grid), we examined a total of 6 conditions: 2 different potential strengths (weak or strong) for each of the 3 types of coupling (attractive, mixed, and repulsive). We computed the $\ell_1$-error $\frac{1}{n} \sum_{s=1}^{n} |p(x_s = 1; \theta) - \widehat{\mu}_s|$, where $\widehat{\mu}_s$ was the approximate marginal computed either by SP or by LD.

| Problem type | | | Method | | | |
| --- | --- | --- | --- | --- | --- | --- |
| | | | Sum-product | | Log-determinant | |
| Graph | Coupling | Strength | Median | Range | Median | Range |
| Full | R | $(0.25, 0.25)$ | 0.035 | $[0.01, 0.10]$ | 0.020 | $[0.01, 0.03]$ |
| | R | $(0.25, 0.50)$ | 0.066 | $[0.03, 0.20]$ | 0.017 | $[0.01, 0.04]$ |
| | M* | $(0.25, 0.25)$ | 0.003 | $[0.00, 0.04]$ | 0.019 | $[0.01, 0.03]$ |
| | M | $(0.25, 0.50)$ | 0.035 | $[0.01, 0.31]$ | 0.010 | $[0.01, 0.06]$ |
| | A* | $(0.25, 0.06)$ | 0.021 | $[0.00, 0.08]$ | 0.026 | $[0.01, 0.06]$ |
| | A | $(0.25, 0.12)$ | 0.422 | $[0.08, 0.86]$ | 0.023 | $[0.01, 0.09]$ |
| Grid | R | $(0.25, 1.0)$ | 0.285 | $[0.04, 0.59]$ | 0.041 | $[0.01, 0.12]$ |
| | R | $(0.25, 2.0)$ | 0.342 | $[0.04, 0.78]$ | 0.033 | $[0.00, 0.12]$ |
| | M* | $(0.25, 1.0)$ | 0.008 | $[0.00, 0.20]$ | 0.016 | $[0.01, 0.02]$ |
| | M | $(0.25, 2.0)$ | 0.053 | $[0.01, 0.54]$ | 0.032 | $[0.01, 0.11]$ |
| | A | $(0.25, 1.0)$ | 0.404 | $[0.06, 0.90]$ | 0.037 | $[0.01, 0.13]$ |
| | A | $(0.25, 2.0)$ | 0.550 | $[0.06, 0.94]$ | 0.031 | $[0.00, 0.12]$ |

**Table 1.** Statistics of the $\ell_1$-approximation error for the sum-product (SP) and log-determinant (LD) methods for the fully connected graph $K_{16}$, as well as the 4-nearest neighbor grid with 16 nodes, with varying coupling and potential strengths.

Table 1 shows quantitative results for 100 trials performed in each of the 12 experimental conditions, including only those trials for which SP converged. The potential strength is given as the pair $(d_{\text{obs}}, d_{\text{coup}})$; note that $d_{\text{obs}} = 0.25$ in all trials. For each method, we show the sample median, and the range [min, max] of the errors. Overall, the performance of LD is better than that of SP , and often substantially so. The performance of SP is slightly better in the regime of weak coupling and relatively strong observations ($\theta_s$ values); see the entries marked with * in the table. In the remaining cases, the LD method outperforms SP, and with a large margin for many examples with strong coupling. The two methods also differ substantially in the ranges of the approximation error. The SP method exhibits some instability, with the error for certain problems being larger than $0.5$; for the same problems, the LD error ranges are much smaller, with a worst case maximum error over all trials and conditions of $0.13$. In addition, the behavior of SP can change dramatically between the weakly coupled and strongly coupled conditions, whereas the LD results remain stable.

## 5 Discussion

In this paper, we developed a new method for approximate inference based on the combination of a Gaussian entropy bound with semidefinite constraints on the marginal polytope. The resultant log-determinant maximization problem can be solved by efficient interior point methods [8]. In experimental trials, the log-determinant method was either comparable or better than the sum-product algorithm, and by a substantial margin for certain problem classes. Of particular interest is that, in contrast to the sum-product algorithm, the performance degrades gracefully as the interaction strength is increased. It can be shown [11, 10] that in the zero-temperature limit, the log-determinant relaxation (12) reduces to a class of semidefinite relaxations that are widely used in combinatorial optimization. One open question is whether techniques for bounding the performance of such semidefinite relaxations [e.g., 3] can be adapted to the finite temperature case.

Although this paper focused exclusively on the binary problem, the methods described here can be extended to other classes of random variables. It remains to develop a deeper understanding of the interaction between the two components to these approximations (i.e., the entropy bound, and the outer bound on the marginal polytope), as well as how to tailor approximations to particular graph structures. Finally, semidefinite constraints can be combined with entropy approximations (preferably convex) other than the Gaussian bound used in this paper, among them "convexified" Bethe/Kikuchi entropy approximations [9].

**Acknowledgements:** Thanks to Constantine Caramanis and Laurent El Ghaoui for helpful discussions. Work funded by NSF grant IIS-9988642, ARO MURI DAA19-02-1-0383, and a grant from Intel Corporation.

## Footnotes

[1]When $\phi_\alpha$ corresponds to an indicator function, then $\mu_\alpha$ is a marginal probability; otherwise, this choice entails a minor abuse of terminology.

[2]To be explicit, letting $\mathbf{z} = (1, \mathbf{x})$, then for any vector $a \in \mathbb{R}^{n+1}$, we have $a^T M_1[\mu] a = a^T \mathbb{E}[\mathbf{z}\mathbf{z}^T] a = \mathbb{E}[\|a^T \mathbf{z}\|^2]$, which is certainly non-negative.

[3]The notation $\mathcal{U}[a, b]$ denotes the uniform distribution on the interval $[a, b]$.

[4] Here $\mathcal{U}[a, b]$ denotes the uniform distribution on $[a, b]$.

[5]More precisely, we updated messages in the log domain as $\gamma \log M_{st}^{new} + (1 - \gamma) \log M_{st}^{old}$.

## References

[1] R. G. Cowell, A. P. Dawid, S. L. Lauritzen, and D. J. Spiegelhalter. *Probabilistic networks and expert systems*. Statistics for Engineering and Information Science. Springer-Verlag, 1999.

[2] M. Deza and M. Laurent. *Geometry of cuts and metric embeddings*. Springer-Verlag, New York, 1997.

[3] M. X. Goemans and D. P. Williamson. Improved approximation algorithms for maximum cut and satisfiability problems using semidefinite programming. *Journal of the ACM*, 42:1115–1145, 1995.

[4] M. Jordan, editor. *Learning in graphical models*. MIT Press, Cambridge, MA, 1999.

[5] J. B. Lasserre. Global optimization with polynomials and the problem of moments. *SIAM Journal on Optimization*, 11(3):796–817, 2001.

[6] R. J. McEliece and M. Yildirim. Belief propagation on partially ordered sets. In D. Gilliam and J. Rosenthal, editors, *Mathematical Theory of Systems and Networks*. Institute for Mathematics and its Applications, 2002.

[7] G. Rockafellar. *Convex Analysis*. Princeton University Press, Princeton, 1970.

[8] L. Vandenberghe, S. Boyd, and S. Wu. Determinant maximization with linear matrix inequality constraints. *SIAM Journal on Matrix Analysis and Applications*, 19:499–533, 1998.

[9] M. J. Wainwright, T. S. Jaakkola, and A. S. Willsky. A new class of upper bounds on the log partition function. In *Uncertainty in Artificial Intelligence*, volume 18, pages 536–543, August 2002.

[10] M. J. Wainwright and M. I. Jordan. Graphical models, exponential families, and variational inference. Technical report, UC Berkeley, Department of Statistics, No. 649, 2003.

[11] M. J. Wainwright and M. I. Jordan. Semidefinite relaxations for approximate inference on graphs with cycles. Technical report, UC Berkeley, UCB/CSD-3-1226, January 2003.

[12] J. S. Yedidia, W. T. Freeman, and Y. Weiss. Understanding belief propagation and its generalizations. Technical Report TR2001-22, Mitsubishi Electric Research Labs, January 2002.